# A latent factor model for highly multi-relational data

**Rodolphe Jenatton**
CMAP, UMR CNRS 7641,
Ecole Polytechnique, Palaiseau, France
jenatton@cmap.polytechnique.fr

**Nicolas Le Roux**
INRIA - SIERRA Project Team,
Ecole Normale Supérieure, Paris, France
nicolas@le-roux.name

**Antoine Bordes**
Heudiasyc, UMR CNRS 7253,
Université de Technologie de Compiègne, France
antoine.bordes@utc.fr

**Guillaume Obozinski**
INRIA - SIERRA Project Team,
Ecole Normale Supérieure, Paris, France
guillaume.obozinski@ens.fr

## Abstract

Many data such as social networks, movie preferences or knowledge bases are multi-relational, in that they describe multiple relations between entities. While there is a large body of work focused on modeling these data, modeling these multiple types of relations jointly remains challenging. Further, existing approaches tend to breakdown when the number of these types grows. In this paper, we propose a method for modeling large multi-relational datasets, with possibly thousands of relations. Our model is based on a bilinear structure, which captures various orders of interaction of the data, and also shares sparse latent factors across different relations. We illustrate the performance of our approach on standard tensor-factorization datasets where we attain, or outperform, state-of-the-art results. Finally, a NLP application demonstrates our scalability and the ability of our model to learn efficient and semantically meaningful verb representations.

## 1 Introduction

Statistical Relational Learning (SRL) [7] aims at modeling data consisting of relations between entities. Social networks, preference data from recommender systems, relational databases used for the semantic web or in bioinformatics, illustrate the diversity of applications in which such modeling has a potential impact.

Relational data typically involve different types of relations between entities or attributes. These entities can be users in the case of social networks or recommender systems, words in the case of lexical knowledge bases, or genes and proteins in the case of bioinformatics ontologies, to name a few. For binary relations, the data is naturally represented as a so called *multi-relational* graph consisting of nodes associated with entities and of different types of edges between nodes corresponding to the different types of relations. Equivalently the data consists of a collection of triplets of the form (subject, relation, object), listing the actual *relationships* where we will call subject and object respectively the first and second term of a binary relation. Relational data typically cumulates many difficulties. First, a large number of relation types, some being significantly more represented than others and possibly concerning only subsets of entities; second, the data is typically noisy and incomplete (missing or incorrect relationships, redundant entities); finally most datasets are large scale with up to millions of entities and billions of links for real-world knowledge bases.

Besides relational databases, SRL can also be used to model natural language semantics. A standard way of representing the meaning of language is to identify entities and relations in texts or speech utterances and to organize them. This can be conducted at various scales, from the word or sentence level (e.g. in parsing or semantic role labeling) to a collection of texts (e.g. in knowledge extraction).

SRL systems are a useful tool there, as they can automatically extract high level information from the collected data by building summaries [22], sense categorization lexicons [11], ontologies [20], etc. Progress in SRL would be likely to lead to advances in natural language understanding.

In this paper, we introduce a model for relational data and apply it to multi-relational graphs and to natural language. In assigning high probabilities to valid relations and low probabilities to all the others, this model extracts meaningful representations of the various entities and relations in the data. Unlike other factorization methods (e.g. [15]), our model is probabilistic which has the advantage of accounting explicitly for the uncertainties in the data. Besides, thanks to a sparse distributed representation of relation types, our model can handle data with a significantly larger number of relation types than was considered so far in the literature (a crucial aspect for natural language data). We empirically show that this approach ties or beats state-of-the-art algorithms on various benchmarks of link prediction, a standard test-bed for SRL methods.

## 2   Related work

A branch of relational learning, motivated by applications such as collaborative filtering and link prediction in networks, models relations between entities as resulting from *intrinsic latent attributes* of these entities.[1] Work in what we will call *relational learning from latent attributes* (RLA) focused mostly on the problem of modeling a single relation type as opposed to trying to model simultaneously a collection of relations which can themselves be similar. As reflected by several formalisms proposed for relational learning [7], it is the latter *multi-relational learning* problem which is needed to model efficiently large scale relational databases. The fact that relations can be similar or related suggests that a superposition of independently learned models for each relation would be highly inefficient especially since the relationships observed for each relation are extremely sparse.

RLA translates often into learning an embedding of the entities, which corresponds algebraically to a matrix factorization problem (typically the matrix of observed relationships). A natural extension to learning multiple relations consists in stacking the matrices to be factorized and applying classical tensor factorization methods such as CANDECOMP/PARAFAC [25, 8]. This approach, which induces inherently some sharing of parameters between both different terms and different relations, has been applied successfully [8] and has inspired some probabilistic formulations [4].

Another natural extension to learning several relations simultaneously can be to share the *common* embedding or the entities across relations via *collective matrix factorization* as proposed in RESCAL [15] and other related work [18, 23].

The simplest form of latent attribute that can be associated to an entity is a latent class: the resulting model is the classical *stochastic blockmodel* [26, 17]. Several clustering-based approaches have been proposed for multi-relational learning: [9] considered a non-parametric Bayesian extension of the *stochastic blockmodel* allowing to automatically infer the number of latent clusters; [14, 28] refined this to allow entities to have a mixed clusters membership; [10] introduced clustering in Markov-Logic networks; [24] used a non-parametric Bayesian clustering of entities embedding in a *collective matrix factorization* formulation. To share parameters between relations, [9, 24, 14, 28] and [10] build models that cluster not only entities but relations as well.

With the same aim of reducing the number of parameters, the Semantic Matching Energy model (SME) of [2] embeds relations as a vector from the same space as the entities and models likely relationships by an energy combining together binary interactions between the relation vector and each of the vectors encoding the two terms.

In terms of scalability, RESCAL [15], which has been shown to achieve state of the art performance on several relation datasets, has recently been applied to the knowledge base YAGO [16] thereby showing its ability to scale well on data with very large numbers of entities, although the number of relations modeled remained moderate (less than 100). As for SME [2], its modeling of relations by vectors allowed it to scale to several thousands of relations. Scalability can be also an issue for nonparametric Bayesian models (e.g. [9, 24]) because of the cost of inference.

# 3 Relational data modeling

We consider relational data consisting of triplets that encode the existence of relation between two entities that we will call the subject and the object. Specifically, we consider a set of $n_s$ subjects $\{\mathcal{S}_i\}_{i \in [\![1;n_s]\!]}$ along with $n_o$ objects $\{\mathcal{O}_k\}_{k \in [\![1;n_o]\!]}$ which are related by some of $n_r$ relations $\{\mathcal{R}_j\}_{j \in [\![1;n_r]\!]}$. A triplet encodes that the relation $\mathcal{R}_j$ holds between the subject $\mathcal{S}_i$ and the object $\mathcal{O}_k$, which we will write $\mathcal{R}_j(\mathcal{S}_i, \mathcal{O}_k) = 1$. We will therefore refer to a triplet also as a *relationship*. A typical example which we will discuss in greater detail is in natural language processing where a triplet $(\mathcal{S}_i, \mathcal{R}_j, \mathcal{O}_k)$ corresponds to the association of a subject and a direct object through a transitive verb. The goal is to learn a model of the relations to reliably predict unseen triplets. For instance, one might be interested in finding a likely relation $\mathcal{R}_j$ based only on the subject and object $(\mathcal{S}_i, \mathcal{O}_k)$.

# 4 Model description

In this work, we formulate the problem of learning a relation as a matrix factorization problem. Following a rationale underlying several previous approaches [15, 24], we consider a model in which entities are embedded in $\mathbb{R}^p$ and relations are encoded as bilinear operators on the entities. More precisely, we assume that the $n_s$ subjects (resp. $n_o$ objects) are represented by vectors of $\mathbb{R}^p$, stored as the columns of the matrix $\mathbf{S} \triangleq [\mathbf{s}^1, \ldots, \mathbf{s}^{n_s}] \in \mathbb{R}^{p \times n_s}$ (resp. as the columns of $\mathbf{O} \triangleq [\mathbf{o}^1, \ldots, \mathbf{o}^{n_o}] \in \mathbb{R}^{p \times n_o}$). Each of the $p$-dimensional representations $\mathbf{s}^i, \mathbf{o}^k$ will have to be learned. The relations are represented by a collection of matrices $(\mathbf{R}_j)_{1 \leq j \leq n_r}$, with $\mathbf{R}_j \in \mathbb{R}^{p \times p}$, which together form a three-dimensional tensor.

We consider a model of the probability of the event $\{\mathcal{R}_j(\mathcal{S}_i, \mathcal{O}_k) = 1\}$. Assuming first that $\mathbf{s}^i$ and $\mathbf{o}^k$ are fixed, our model is derived from a logistic model $\mathbb{P}[\mathcal{R}_j(\mathcal{S}_i, \mathcal{O}_k) = 1] \triangleq \sigma\big(\eta_{ik}^{(j)}\big)$, with $\sigma(t) \triangleq 1/(1 + e^{-t})$. A natural form for $\eta_{ik}^{(j)}$ is a linear function of the tensor product $\mathbf{s}^i \otimes \mathbf{o}^k$ which we can write $\eta_{ik}^{(j)} = \langle \mathbf{s}^i, \mathbf{R}_j \mathbf{o}^k \rangle$ where $\langle \cdot, \cdot \rangle$ is the usual inner product in $\mathbb{R}^p$. If we think now of learning $\mathbf{s}^i$, $\mathbf{R}_j$ and $\mathbf{o}^k$ for all $(i, j, k)$ simultaneously, this model learns together the matrices $\mathbf{R}_j$ and optimal embeddings $\mathbf{s}^i, \mathbf{o}^k$ of the entities so that the usual logistic regressions based on $\mathbf{s}^i \otimes \mathbf{o}^k$ predict well the probability of the observed relationships. This is the initial model considered in [24] and it matches the model considered in [16] if the least-square loss is substituted to the logistic loss. We will refine this model in two ways: first by redefining the term $\eta_{ik}^{(j)}$ as a function $\eta_{ik}^{(j)} \triangleq \mathcal{E}(\mathbf{s}^i, \mathbf{R}_j, \mathbf{o}^k)$ taking into account the different orders of interactions between $\mathbf{s}^i$, $\mathbf{o}^k$ and $\mathbf{R}_j$, second, by parameterizing the relations $\mathbf{R}_j$ by latent "relational" factors that reduce the overall number of parameters of the model.

## 4.1 A multiple order log-odds ratio model

One way of thinking about the probability of occurrence of a specific relationship corresponding to the triplet $(\mathcal{S}_i, \mathcal{R}_j, \mathcal{O}_k)$ is as resulting (a) from the marginal propensity of individual entities $\mathcal{S}_i, \mathcal{O}_k$ to enter relations and the marginal propensity of relations $\mathcal{R}_j$ to occur, (b) from 2-way interactions of $(\mathcal{S}_i, \mathcal{R}_j)$, $(\mathcal{R}_j, \mathcal{O}_k)$ corresponding to entities tending to occur marginally as left of right terms of a relation (c) from 2-way interactions of pairs of entities $(\mathcal{S}_i, \mathcal{O}_k)$ that overall tend to have more relations together, and (d) the 3-way dependencies between $(\mathcal{S}_i, \mathcal{R}_j, \mathcal{O}_k)$.

In NLP, we often refer to these as respectively unigram, bigram and trigram terms, a terminology which we will reuse in the rest of the paper. We therefore design $\mathcal{E}(\mathbf{s}^i, \mathbf{R}_j, \mathbf{o}^k)$ to account for these interactions of various orders, retaining only [2] terms involving $\mathbf{R}_j$.

In particular, introducing new parameters $\mathbf{y}, \mathbf{y}', \mathbf{z}, \mathbf{z}' \in \mathbb{R}^p$, we define $\eta_{ik}^{(j)} = \mathcal{E}(\mathbf{s}^i, \mathbf{R}_j, \mathbf{o}^k)$ as

$$\mathcal{E}(\mathbf{s}^i, \mathbf{R}_j, \mathbf{o}^k) \triangleq \langle \mathbf{y}, \mathbf{R}_j \mathbf{y}' \rangle + \langle \mathbf{s}^i, \mathbf{R}_j \mathbf{z} \rangle + \langle \mathbf{z}', \mathbf{R}_j \mathbf{o}^k \rangle + \langle \mathbf{s}^i, \mathbf{R}_j \mathbf{o}^k \rangle, \qquad (1)$$

where $\langle \mathbf{y}, \mathbf{R}_j \mathbf{y}' \rangle$, $\langle \mathbf{s}^i, \mathbf{R}_j \mathbf{z} \rangle + \langle \mathbf{z}', \mathbf{R}_j \mathbf{o}^k \rangle$ and $\langle \mathbf{s}^i, \mathbf{R}_j \mathbf{o}^k \rangle$ are the uni-, bi- and trigram terms. This parametrization is redundant in general given that $\mathcal{E}(\mathbf{s}^i, \mathbf{R}_j, \mathbf{o}^k)$ is of the form $\langle (\mathbf{s}^i + \mathbf{z}), \mathbf{R}_j (\mathbf{o}^k + \mathbf{z}') \rangle + b_j$; but it is however useful in the context of a regularized model (see Section 5).

### 4.2 Sharing parameters across relations through latent factors

When learning a large number of relations, the number of observations for many relations can be quite small, leading to a risk of overfitting. Sutskever et al. [24] addressed this issue with a nonparametric Bayesian model inducing clustering of both relations and entities. SME [2] proposed to embed relations as vectors of $\mathbb{R}^p$, like entities, to tackle problems with hundreds of relation types.

With a similar motivation to decrease the overall number of parameters, instead of using a general parameterization of the matrices $\mathbf{R}_j$ as in RESCAL [16], we require that all $\mathbf{R}_j$ decompose over a common set of $d$ rank one matrices $\{\mathbf{\Theta}_r\}_{1 \leq r \leq d}$ representing some canonical relations:

$$\mathbf{R}_j = \sum_{r=1}^d \boldsymbol{\alpha}_r^j \mathbf{\Theta}_r, \quad \text{for some sparse } \boldsymbol{\alpha}^j \in \mathbb{R}^d \quad \text{and} \quad \mathbf{\Theta}_r = \mathbf{u}_r \mathbf{v}_r^\top \quad \text{for } \mathbf{u}_r, \mathbf{v}_r \in \mathbb{R}^p. \quad (2)$$

The combined effect of (a) the sparsity of the decomposition and (b) the fact that $d \ll n_r$ leads to sharing parameters across relations. Further, constraining $\mathbf{\Theta}_r$ to be the outer product $\mathbf{u}_r \mathbf{v}_r^\top$ also speeds up all computations relying on linear algebra.

## 5 Regularized formulation and optimization

Denoting $\mathcal{P}$ (resp. $\mathcal{N}$) the set of indices of positively (resp. negatively) labeled relations, the likelihood we seek to maximize is

$$\mathcal{L} \triangleq \prod_{(i,j,k) \in \mathcal{P}} \mathbb{P}[\mathcal{R}_j(\mathcal{S}_i, \mathcal{O}_k) = 1] \quad \cdot \quad \prod_{(i',j',k') \in \mathcal{N}} \mathbb{P}[\mathcal{R}_{j'}(\mathcal{S}_{i'}, \mathcal{O}_{k'}) = 0].$$

The log-likelihood is thus $\log(\mathcal{L}) = \sum_{(i,j,k) \in \mathcal{P}} \eta_{ik}^{(j)} - \sum_{(i,j,k) \in \mathcal{P} \cup \mathcal{N}} \log(1 + \exp(\eta_{ik}^{(j)}))$, with $\eta_{ik}^{(j)} = \mathcal{E}(\mathbf{s}^i, \mathbf{R}_j, \mathbf{o}^k)$. To properly normalize the terms appearing in (1) and (2), we carry out the minimization of the negative log-likelihood over a specific constraint set, namely

$$\min_{\substack{\mathbf{S},\mathbf{O},\{\boldsymbol{\alpha}^j\}, \\ \{\mathbf{\Theta}_r\}\mathbf{y},\mathbf{y}',\mathbf{z},\mathbf{z}'}} - \log(\mathcal{L}), \quad \text{with} \quad \begin{cases} \|\boldsymbol{\alpha}^j\|_1 \leq \lambda, \ \mathbf{\Theta}_r = \mathbf{u}_r \cdot \mathbf{v}_r^\top, \\ \mathbf{z} = \mathbf{z}', \mathbf{O} = \mathbf{S}, \\ \mathbf{s}^j, \mathbf{o}^k, \mathbf{y}, \mathbf{y}', \mathbf{z}, \mathbf{u}_r \text{ and } \mathbf{v}_r \text{ in the ball } \{\mathbf{w}; \|\mathbf{w}\|_2 \leq 1\}. \end{cases}$$

We chose to constrain $\boldsymbol{\alpha}$ in $\ell_1$-norm based on preliminary experiments suggesting that it led to better results that the regularization in $\ell_2$-norm. The regularization parameter $\lambda \geq 0$ controls the sparsity of the relation representations in (2). The equality constraints induce a shared representations between subject and objects which were shown to improve the model in preliminary experiments. Given the fact that the model is conditional on a pair $(\mathbf{s}^i, \mathbf{o}^k)$, only a single scale parameter, namely $\boldsymbol{\alpha}_r^j$, is necessary in the product $\alpha_r^j \langle \mathbf{s}^i, \mathbf{\Theta}_r \mathbf{o}^k \rangle$, which motivates all the Euclidean unit ball constraints.

### 5.1 Algorithmic approach

Given the large scale of the problems we are interested in (e.g., $|\mathcal{P}| \approx 10^6$), and since we can project efficiently onto the constraint set (both the projections onto $\ell_1$- and $\ell_2$-norm balls can be performed in linear time [1]), our optimization problem lends itself well to a stochastic projected gradient algorithm [3].

In order to speed up the optimization, we use several practical tricks. First, we consider a stochastic gradient descent scheme with mini-batches containing 100 triplets. Second, we use stepsizes of the form $a/(1 + k)$ with $k$ the iteration number and $a$ a scalar (common to all parameters) optimized over a logarithmic grid on a validation set.[3]

Additionally, we cannot treat the NLP application (see Sec. 8) as a standard tensor factorization problem. Indeed, in that case, we only have access to the positively labeled triplets $\mathcal{P}$. Following [2], we generate elements in $\mathcal{N}$ by considering triplets of the form $\{(i, j', k)\}, j' \neq j$ for each $(i, j, k) \in \mathcal{P}$. In practice, for each positive triplet, we sample a number of artificial negative triplets containing the same subject and object as our positive triplet but different verbs. This allowed us to change

the problem into a multiclass one where the goal was to correctly classify the "positive" verb, in competition with the "negative" ones.

The standard approach for this problem is to use a multinomial logistic function. However, such a function is highly sensitive to the particular choice of negative verbs and using all the verbs as negative ones would be too costly. Another more robust approach consists in using the likelihood function defined above where we try to classify the positive verbs as a valid relationship and the negative ones as invalid relationships. Further, this approximation to the multinomial logistic function is asymptotically unbiased.

Finally, we observed that it was advantageous to down-weight the influence of the negative verbs to avoid swamping the influence of the positive ones.

## 6    Relation to other models

Our model is closely related to several other models. First, if $d$ is large, the parameters of the $\mathbf{R}_j$ are decoupled and the RESCAL model is retrieved (up to a change of loss function).

Second, our model is also related to classical tensor factorization model such as PARAFAC which approximate the tensor $[\mathcal{R}_k(\mathcal{S}_i, \mathcal{O}_j)]_{i,j,k}$ in the least-square sense by a low rank tensor $\tilde{\mathbf{H}}$ of the form $\sum_{r=1}^d \boldsymbol{\alpha}_r \otimes \boldsymbol{\beta}_r \otimes \boldsymbol{\gamma}_r$ for $(\boldsymbol{\alpha}_r, \boldsymbol{\beta}_r, \boldsymbol{\gamma}_r) \in \mathbb{R}^{n_r \times n_s \times n_o}$. The parameterization of all $\mathbf{R}_j$ as linear combinations of $d$ rank one matrices is in fact equivalent to constraining the tensor $\mathbf{R} = \{\mathbf{R}_j\}_{j \in [\![1;n_r]\!]}$ to be the low rank tensor $\mathbf{R} = \sum_{r=1}^d \boldsymbol{\alpha}_r \otimes \mathbf{u}_r \otimes \mathbf{v}_r$. As a consequence, the tensor of all trigram terms[4] can be written also as $\sum_{r=1}^d \boldsymbol{\alpha}_r \otimes \boldsymbol{\beta}_r \otimes \boldsymbol{\gamma}_r$ with $\boldsymbol{\beta}_r = \mathbf{S}^\top \mathbf{u}_r$ and $\boldsymbol{\gamma}_r = \mathbf{O}^\top \mathbf{v}_r$. This shows that our model is a particular form of tensor factorization which reduces to PARAFAC (up to a change of loss function) when $p$ is sufficiently large.

Finally, the approach considered in [2] seems a priori quite different from ours, in particular since relations are in that work embedded as vectors of $\mathbb{R}^p$ like the entities as opposed to matrices of $\mathbb{R}^{p \times p}$ in our case. This choice can be detrimental to model complex relation patterns as we show in Section 7. In addition, no parameterization of the model [2] is able of handling both bigram and trigram interactions as we propose.

## 7    Application to multi-relational benchmarks

We report in this section the performance of our model evaluated on standard tensor-factorization datasets, which we first briefly describe.

### 7.1    Datasets

**Kinships.** Australian tribes are renowned among anthropologists for the complex relational structure of their kinship systems. This dataset, created by [6], focuses on the Alyawarra, a tribe from Central Australia. 104 tribe members were asked to provide the kinship terms they used for one another. This results in graph of 104 entities and 26 relation types, each of them depicting a different kinship term, such as Adiadya or Umbaidya. See [6] or [9] for more details.

**UMLS.** This dataset contains data from the Unified Medical Language System semantic work gathered by [12]. This consists in a graph with 135 entities and 49 relation types. The entities are high-level concepts like 'Disease or Syndrome', 'Diagnostic Procedure', or 'Mammal'. The relations represent verbs depicting causal influence between concepts like 'affect' or 'cause'.

**Nations.** This dataset groups 14 countries (Brazil, China, Egypt, etc.) with 56 binary relation types representing interactions among them like 'economic_aid', 'treaties' or 'rel_diplomacy', and 111 features describing each country, which we treated as 111 additional entities interacting with the country through an additional 'has_feature' relation[5]. See [21] for details.

| Datasets | | Our approach | RESCAL [16] | MRC [10] | SME [2] |
|---|---|---|---|---|---|
| Kinships | *Area under PR curve* | **0.946** ± 0.005 | **0.95** | 0.84 | 0.907 ± 0.008 |
| | *Log-likelihood* | **-0.029** ± 0.001 | N/A | -0.045 ± 0.002 | N/A |
| UMLS | *Area under PR curve* | **0.990** ± 0.003 | 0.98 | 0.98 | 0.983 ± 0.003 |
| | *Log-likelihood* | **-0.002** ± 0.0003 | N/A | -0.004 ± 0.001 | N/A |
| Nations | *Area under PR curve* | **0.909** ± 0.009 | 0.84 | 0.75 | **0.883** ± 0.02 |
| | *Log-likelihood* | **-0.202** ± 0.008 | N/A | -0.311 ± 0.022 | N/A |

Table 1: Comparisons of the performance obtained by our approach, RESCAL [16], MRC [10] and SME [2] over three standard datasets. The results are computed by 10-fold cross-validation.

## 7.2 Results

These three datasets are relatively small-scale and contain only a few relationships (in the order of tens). Since our model is primarily designed to handle a large number of relationships (see Sec. 4.2), this setting is the most favorable to evaluate the potential of our approach. As reported in Table 1, our method does nonetheless yield better or equally good performance as previous state-of-the-art techniques, both in terms of area under the precision-recall curve (AUC) and log-likelihood (LL). The results displayed in Table 1 are computed by 10-fold cross-validation[6], averaged over 10 random splits of the datasets (90% for cross-validation and 10% for testing). We chose to compare our model with RESCAL [16], MRC [10] and SME [2] because they achieved the best published results on these benchmarks in terms of AUC and LL, to the best of our knowledge.

Interestingly, the trigram term from (1) is essential to obtain good performance on Kinships (with the trigram term removed, we obtain $0.16$ in AUC and $-0.14$ in LL), thus showing the need for modeling 3-way interactions in complex relational data. Moreover, and as expected due to the low number of relations, the value of $\lambda$ selected by cross-validation is quite large ($\lambda = n_r \times d$), and as consequence does not lead to sparsity in (2). Results on this dataset also exhibits the benefit of modeling relations with matrices instead of vectors as does SME [2].

Zhu [28] recently reported results on Nations and Kinships evaluated in terms of area under the receiver-operating-characteristic curve instead of area under the precision-recall curve as we display in Table 1. With this other metric, our model obtains $0.953$ on Nations and $0.992$ on Kinships and hence outperforms Zhu's approach, which achieves $0.926$ and $0.962$ respectively.

## 8 Learning semantic representations of verbs

By providing an approach to model the relational structure of language, SRL can be of great use for learning natural language semantics. Hence, this section proposes an application of our method on text data from Wikipedia for learning a representation of words, with a focus on verbs.

### 8.1 Experimental setting

**Data.** We collected this data in two stages. First, the SENNA software[7] [5] was used to perform part-of-speech tagging, chunking, lemmatization[8] and semantic role labeling on $\approx$2,000,000 Wikipedia articles. This data was then filtered to only select sentences for which the syntactic structure was (subject, verb, direct object) with each term of the triplet being a single word from the WordNet lexicon [13]. Subjects and direct objects ended up being all single nouns, whose dictionary size is 30,605. The total number of relations in this dataset (i.e. the number of verbs) is 4,547: this is much larger than for previously published multi-relational benchmarks. We kept 1,000,000 such relationships to build a training set, 50,000 for a validation set and 250,000 for test. All triplets are unique and we made sure that all words appearing in the validation or test sets were occurring in the training set.[9]

| | synonyms not considered | | | best synonyms considered | | |
|---|---|---|---|---|---|---|
| | median/mean rank | p@5 | p@20 | median/mean rank | p@5 | p@20 |
| Our approach | 50 / **195.0** | **0.78** | **0.95** | 19 / **96.7** | **0.89** | **0.98** |
| SME [2] | 56 / 199.6 | 0.77 | **0.95** | 19 / 99.2 | **0.89** | **0.98** |
| Bigram | **48** / 517.4 | 0.72 | 0.83 | **17** / 157.7 | 0.87 | 0.95 |

Table 2: Performance obtained on the NLP dataset by our approach, SME [2] and a bigram model. Details about the statistics of the table are given in the text.

**Practical training setup.** During the training phase, we optimized over the validation set various parameters, namely, the size $p \in \{25, 50, 100\}$ of the representations, the dimension $d \in \{50, 100, 200\}$ of the latent decompositions (2), the value of the regularization parameter $\lambda$ as a fraction $\{1, 0.5, 0.1, 0.05, 0.01\}$ of $n_r \times d$, the stepsize in $\{0.1, 0.05, 0.01\}$ and the weighting of the negative triplets. Moreover, to speed up the training, we gradually increased the number of sampled negative verbs (cf. Section 5.1), from 25 up to 50, which had the effect of refining the training.

## 8.2  Results

**Verb prediction.** We first consider a direct evaluation of our approach based on the test set of 250,000 instances by measuring how well we predict a relevant and meaningful verb given a pair (subject, direct object). To this end, for each test relationship, we rank all verbs using our probability estimates given a pair (subject, direct object). Table 2 displays our results with two kinds of metrics, namely, (1) the rank of the correct verb and (2) the fraction of test examples for which the correct verb is ranked in the top $z\%$ of the list. The latter criterion is referred to as p@$z$. In order to evaluate if some language semantics is captured by the representations, we also consider a less conservative approach where, instead of focusing on the correct verb only, we measure the minimum rank achieved over its set of synonyms obtained from WordNet. Our method is compared with that of SME [2], which was shown to scale well on data with large sets of relations, and with a bigram model, which estimates the probabilities of the pairs (subject, verb) and (verb, direct object).

The first observation is that the task of verb prediction can be quite well addressed by a simple model based on 2-way interactions, as shown by the good median rank obtained by the bigram model. This is confirmed by the mild influence of the trigram term on the performance of our model. On this data, we experienced that using bigram interactions in our energy function was essential to achieve good predictions. However, the drop in the mean rank between our approach and the bigram-only model still indicates that many examples do need a richer model to be correctly handled. By comparison, we tend to consistently match or improve upon the performance of SME. Remarkably, model selection led to the choice of $\lambda = 0.1 \cdot n_r \times d$ for which the coefficients $\boldsymbol{\alpha}$ of the representations (2) are sparse in the sense they are dominated by few large values (e.g., the top 2% of the largest values of $\boldsymbol{\alpha}$ account for about 25% of the total $\ell_1$-norm $\|\boldsymbol{\alpha}\|_1$).

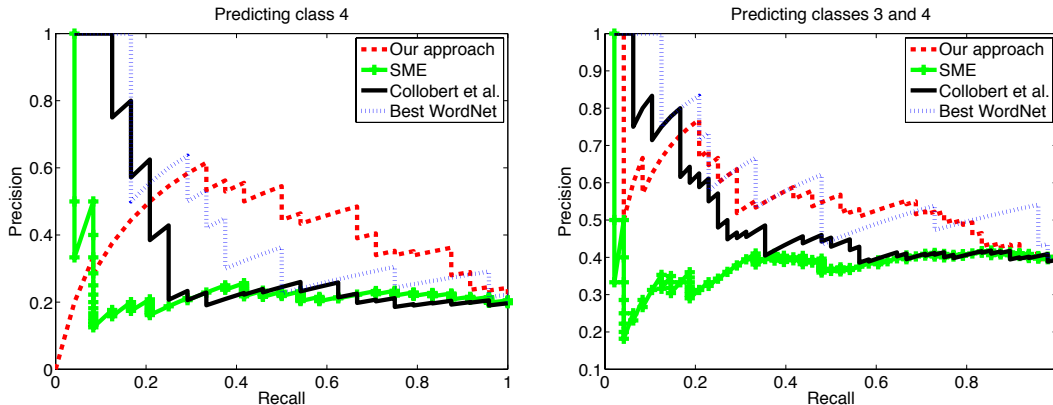

Figure 1: Precision-recall curves for the task of lexical similarity classification. The curves are computed based on different similarity measures between verbs, namely, our approach, SME [2], Collobert et al. [5] and the best (out of three) WordNet similarity measure [13]. Details about the task can be found in the text.

|  | Our approach | SME [2] | Collobert et al. [5] | Best WordNet [19] |
|---|---|---|---|---|
| AUC (class 4) | **0.40** | 0.21 | 0.31 | **0.40** |
| AUC (classes 3&4) | 0.54 | 0.36 | 0.48 | **0.59** |

Table 3: Performance obtained on a task of lexical similarity classification [27], where we compare our approach, SME [2], Collobert et al.'s word embeddings [5] and the best (out of 3) WordNet Similarity measure [19] using area under the precision-recall curve. Details are given in the text.

**Lexical similarity classification.** Our method learns latent representations for verbs and imposes them some structure via shared parameters, as shown in Section 4.2. This should lead to similar representations for similar verbs. We consider the task of lexical similarity classification described in [27] to evaluate this hypothesis. Their dataset consists of 130 pairs of verbs labeled by humans with a score in $\{0, 1, 2, 3, 4\}$. Higher scores means a stronger semantic similarity between the verbs composing the pair. For instance, (divide,split) is labeled 4, while (postpone,show) has a score of 0.

Based on the pairwise Euclidean distances[10] between our learned verb representations $\mathbf{R}_j$, we try to predict the class 4 (and also the "merged" classes $\{3, 4\}$) by using the assumption that the smallest the distance between $\mathbf{R}_i$ and $\mathbf{R}_j$, the more likely the pair $(i, j)$ should be labeled as 4. We compare to representations learnt by [2] on the same training data, to word embeddings of [5] (which are considered as efficient features in Natural Language Processing), and with three similarity measures provided by WordNet Similarity [19]. For the latest, we only display the best one, named "path", which is built by counting the number of nodes along the shortest path between the senses in the "is-a" hierarchies of WordNet.

We report our results on precision-recall curves displayed in Figure 1 and the corresponding areas under the curve (AUC) in Table 3. Even though we tend to miss the first few pairs, we compare favorably to [2] and [5] and our AUC is close to the reference established by WordNet Similarity. Our method is capable of encoding meaningful semantic embeddings for verbs, even though it has been trained on noisy, automatically collected data and in spite of the fact that it was not our primary goal that distance in parameter space should satisfy any condition. Performance might be improved by training on cleaner triplets, such as those collected by [11].

# 9  Conclusion

Designing methods capable of handling large amounts of linked relations seems necessary to be able to model the wealth of relations underlying the semantics of any real-world problems. We tackle this problem by using a shared representation of relations naturally suited to multi-relational data, in which entities have a unique representation shared between relation types, and where we propose that relation themselves decompose over latent "relational" factors. This new approach ties or beats state-of-the art models on both standard relational learning problems and an NLP task. The decomposition of relations over latent factors allows a significant reduction of the number of parameters and is motivated both by computational and statistical reasons. In particular, our approach is quite scalable both with respect to the number of relations and to the data samples.

One might wonder about the relative importance of the various terms in our formulation. Interestingly, though the presence of the trigram term was crucial in the tensor factorization problems, it played a marginal role in the NLP experiment, where most of the information was contained in the bigram and unigram terms.

Finally, we believe that exploring the similarities of the relations through an analysis of the latent factors could provide some insight on the structures shared between different relation types.

**Acknowledgments**

This work was partially funded by the Pascal2 European Network of Excellence. NLR and RJ are supported by the European Research Council (resp., SIERRA-ERC-239993 & SIPA-ERC-256919).

## Footnotes

[1]This is called *Statistical Predicate Invention* by [10].

[2] This is motivated by the fact that we are primarily interested in modelling the relations terms, and that it is not necessary to introduce all terms to fully parameterize the model.

[3]The code is available under an open-source license from `http://goo.gl/TGYuh`.

[4]Other terms can be decomposed in a similar way.

[5]The resulting new relationships were only used for training, and not considered at test time.

[6] The values of $\lambda$, $d$ and $p$ are searched in $n_r \times d \cdot \{0.05, 0.1, 0.5, 1\}$, $\{100, 200, 500\}$ and $\{10, 25, 50\}$.

[7] Available from `ronan.collobert.com/senna/`.

[8] Lemmatization was carried out using NLTK (`nltk.org`) and transforms a word into its base form.

[9] The data set is available under an open-source license from `http://goo.gl/TGYuh`.

[10]Other distances could of course be considered, we choose the Euclidean metric for simplicity.

# References

[1] F. Bach, R. Jenatton, J. Mairal, and G. Obozinski. Optimization with sparsity-inducing penalties. *Foundations and Trends in Machine Learning*, 4(1):1–106, 2011.

[2] A. Bordes, X. Glorot, J. Weston, and Y. Bengio. A semantic matching energy function for learning with multi-relational data. *Machine Learning*, 2012. To appear.

[3] L. Bottou and Y. LeCun. Large scale online learning. In *Advances in Neural Information Processing Systems*, volume 16, pages 217–224, 2004.

[4] W. Chu and Z. Ghahramani. Probabilistic models for incomplete multi-dimensional arrays. *Journal of Machine Learning Research - Proceedings Track*, 5:89–96, 2009.

[5] R. Collobert, J. Weston, L. Bottou, M. Karlen, K. Kavukcuoglu, and P. Kuksa. Natural language processing (almost) from scratch. *JMLR*, 12:2493–2537, 2011.

[6] W. Denham. *The detection of patterns in Alyawarra nonverbal behavior*. PhD thesis, 1973.

[7] L. Getoor and B. Taskar. *Introduction to Statistical Relational Learning (Adaptive Computation and Machine Learning)*. The MIT Press, 2007.

[8] R. A. Harshman and M. E. Lundy. Parafac: parallel factor analysis. *Comput. Stat. Data Anal.*, 18(1):39–72, Aug. 1994.

[9] C. Kemp, J. B. Tenenbaum, T. L. Griffiths, T. Yamada, and N. Ueda. Learning systems of concepts with an infinite relational model. In *Proc. of AAAI*, pages 381–388, 2006.

[10] S. Kok and P. Domingos. Statistical predicate invention. In *Proceedings of the 24th international conference on Machine learning*, pages 433–440, 2007.

[11] A. Korhonen, Y. Krymolowski, and T. Briscoe. A large subcategorization lexicon for natural language processing applications. In *Proceedings of LREC*, 2006.

[12] A. T. McCray. An upper level ontology for the biomedical domain. *Comparative and Functional Genomics*, 4:80–88, 2003.

[13] G. Miller. WordNet: a Lexical Database for English. *Communications of the ACM*, 38(11):39–41, 1995.

[14] K. Miller, T. Griffiths, and M. Jordan. Nonparametric latent feature models for link prediction. In *Advances in Neural Information Processing Systems 22*, pages 1276–1284. 2009.

[15] M. Nickel, V. Tresp, and H.-P. Kriegel. A three-way model for collective learning on multi-relational data. In *Proceedings of the 28th Intl Conf. on Mach. Learn.*, pages 809–816, 2011.

[16] M. Nickel, V. Tresp, and H.-P. Kriegel. Factorizing YAGO: scalable machine learning for linked data. In *Proc. of the 21st intl conf. on WWW*, pages 271–280, 2012.

[17] K. Nowicki and T. A. B. Snijders. Estimation and prediction for stochastic blockstructures. *Journal of the American Statistical Association*, 96(455):1077–1087, 2001.

[18] A. Paccanaro and G. Hinton. Learning distributed representations of concepts using linear relational embedding. *IEEE Trans. on Knowl. and Data Eng.*, 13:232–244, 2001.

[19] T. Pedersen, S. Patwardhan, and J. Michelizzi. Wordnet:: Similarity: measuring the relatedness of concepts. In *Demonstration Papers at HLT-NAACL 2004*, pages 38–41, 2004.

[20] H. Poon and P. Domingos. Unsupervised ontology induction from text. In *Proceedings of the 48th Annual Meeting of the Association for Computl Linguistics*, pages 296–305, 2010.

[21] R. J. Rummel. Dimensionality of nations project: Attributes of nations and behavior of nation dyads. In *ICPSR data file*, pages 1950–1965. 1999.

[22] D. Shen, J.-T. Sun, H. Li, Q. Yang, and Z. Chen. Document summarization using conditional random fields. In *Proc. of the 20th Intl Joint Conf. on Artif. Intel.*, pages 2862–2867, 2007.

[23] A. P. Singh and G. J. Gordon. Relational learning via collective matrix factorization. In *Proc. of SIGKDD'08*, pages 650–658, 2008.

[24] I. Sutskever, R. Salakhutdinov, and J. Tenenbaum. Modelling relational data using bayesian clustered tensor factorization. In *Adv. in Neur. Inf. Proc. Syst. 22*, 2009.

[25] L. R. Tucker. Some mathematical notes on three-mode factor analysis. *Psychometrika*, 31:279–311, 1966.

[26] Y. J. Wang and G. Y. Wong. Stochastic blockmodels for directed graphs. *Journal of the American Statistical Association*, 82(397), 1987.

[27] D. Yang and D. M. W. Powers. Verb similarity on the taxonomy of wordnet. *Proceedings of GWC-06*, pages 121–128, 2006.

[28] J. Zhu. Max-margin nonparametric latent feature models for link prediction. In *Proceedings of the 29th Intl Conference on Machine Learning*, 2012.

